# Message Errors in Belief Propagation

**Alexander T. Ihler, John W. Fisher III, and Alan S. Willsky**
Department of Electrical Engineering and Computer Science
Massachusetts Institute of Technology
*ihler@mit.edu, fisher@csail.mit.edu, willsky@mit.edu*

## Abstract

Belief propagation (BP) is an increasingly popular method of performing approximate inference on arbitrary graphical models. At times, even further approximations are required, whether from quantization or other simplified message representations or from stochastic approximation methods. Introducing such errors into the BP message computations has the potential to adversely affect the solution obtained. We analyze this effect with respect to a particular measure of message error, and show bounds on the accumulation of errors in the system. This leads both to convergence conditions and error bounds in traditional and approximate BP message passing.

## 1 Introduction

Graphical models and message-passing algorithms defined on graphs are a growing field of research. In particular, the *belief propagation* (BP, or sum-product) algorithm has become a popular means of solving inference problems exactly or approximately. One part of its appeal is its optimality for tree-structured graphical models (models which contain no loops). However, its is also widely applied to graphical models with cycles. In these cases it may not converge, and if it does its solution is approximate; however in practice these approximations are often good. Recently, further justifications for loopy belief propagation have been developed, including a few convergence results for graphs with cycles [1–3].

The approximate nature of loopy BP is often a more than acceptable price for efficient inference; in fact, it is sometimes desirable to make *additional* approximations. There may be a number of reasons for this—for example, when exact message representation is computationally intractable, the messages may be approximated stochastically [4] or deterministically by discarding low-likelihood states [5]. For BP involving continuous, non-Gaussian potentials, some form of approximation is required to obtain a finite parametrization for the messages [6–8]. Additionally, graph simplification by edge removal may be regarded as a coarse form of message approximation. Finally, one may wish to approximate the messages and reduce their representation size for another reason—to decrease the communications required for distributed inference applications. In a distributed environment, one may approximate the transmitted message to reduce its representational cost [9], or discard it entirely if it is deemed "sufficiently similar" to the previously sent version [10]. This may significantly reduce the amount of communication required.

Given that message approximation may be desirable, we would like to know what effect the introduced errors have on our overall solution. To characterize the effect in graphs

with cycles, we analyze the deviation from a solution given by "exact" loopy BP (*not*, as is typically considered, the deviation of loopy BP from the true marginal distributions). In the process, we also develop some results on the convergence of loopy BP. Section 3 describes the major themes of the paper; but first we provide a brief summary of belief propagation.

## 2 Graphical Models and Belief Propagation

Graphical models provide a convenient means of representing conditional independence relations among large numbers of random variables. Specifically, each node $s$ in a graph is associated with a random variable $x_s$, while the set of edges $\mathcal{E}$ is used to describe the conditional dependency structure of the variables. A distribution satisfies the conditional independence relations specified by an undirected graph if it factors into a product of potential functions $\psi$ defined on the cliques (fully-connected subsets) of the graph; the converse is also true if $p(\mathbf{x})$ is strictly positive [11]. Here we consider graphs with at most pairwise interactions (a typical assumption in BP), where the distribution factors according to

$$p(\mathbf{x}) = \prod_{(s,t)\in\mathcal{E}} \psi_{st}(x_s, x_t) \prod_s \psi_s(x_s) \tag{1}$$

The goal of belief propagation [12], or BP, is to compute the marginal distribution $p(x_t)$ at each node $t$. BP takes the form of a message-passing algorithm between nodes, expressed in terms of an update to the outgoing message from each node $t$ to each neighbor $s$ in terms of the (previous iteration's) incoming messages from $t$'s neighbors $\Gamma_t$,

$$m_{ts}(x_s) \propto \int \psi_{ts}(x_t, x_s)\psi_t(x_t) \prod_{u\in\Gamma_t\backslash s} m_{ut}(x_t)dx_t \tag{2}$$

Typically each message is normalized so as to integrate to unity (and we assume that such normalization is possible). At any iteration, one may calculate the *belief* at node $t$ by

$$M_t^i(x_t) \propto \psi_t(x_t) \prod_{u\in\Gamma_t} m_{ut}^i(x_t) \tag{3}$$

For tree-structured graphical models, belief propagation can be used to efficiently perform exact marginalization. Specifically, the iteration (2) converges in a finite number of iterations (at most the length of the longest path in the graph), after which the belief (3) equals the correct marginal $p(x_t)$. However, as observed by [12], one may also apply belief propagation to arbitrary graphical models by following the same *local* message passing rules at each node and ignoring the presence of cycles in the graph; this procedure is typically referred to as "loopy" BP.

For loopy BP, the sequence of messages defined by (2) is not guaranteed to converge to a fixed point after any number of iterations. Under relatively mild conditions, one may guarantee the existence of fixed points [13]. However, they may not be unique, nor are the results exact (the belief $M_t^i$ does not converge to the true marginal). In practice however the procedure often arrives at a reasonable set of approximations to the correct marginal distributions.

It is sometimes convenient to think of loopy BP in terms of its *computation tree* [2]. The $n$-level computation tree rooted at some node $t$ is a tree-

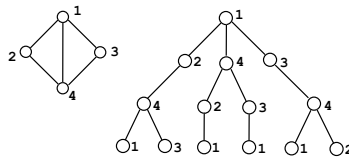

Figure 1: For a graph with cycles, one may show an equivalence between $n$ iterations of loopy BP and the $n$-level computation tree (shown here for $n = 3$ and rooted at node 1; example from [2]).

structured "unrolling" of the graph, so that $n$ iterations of loopy BP on the original graph is equivalent at the node $t$ to exact inference on the computation tree. An example of this structure is shown in Figure 1.

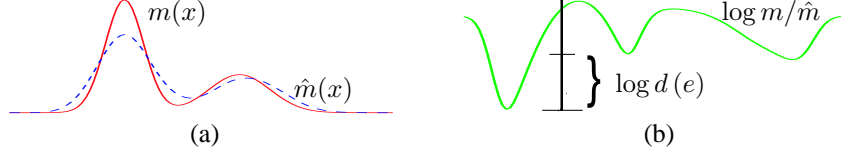

(a)  (b)

Figure 2: (a) A message $m(x)$, solid, and its approximation $\hat{m}(x)$, dashed. (b) Their log-ratio $\log m(x)/\hat{m}(x)$; $\log d(e)$ characterizes their similarity by measuring the error's dynamic range.

## 3  Overview of Results

To orient the reader, we lay out the order and general results which are obtained in this paper. We begin by considering multiplicative error functions which describe the difference between a "true" message $m(x)$ (typically meaning consistent with some BP fixed-point) and some approximation $\hat{m}(x) = m(x) \cdot e(x)$. We apply a particular functional measure $d(e)$ (defined below) and show how this measure behaves with respect to the BP equations (2) and (3). When applied to traditional BP, this results in a novel sufficient condition for its convergence to a unique solution, specifically

$$\max_{(s,t)\in\mathcal{E}} \sum_{u\in\Gamma_t\setminus s} \frac{d(\psi_{ut})^2 - 1}{d(\psi_{ut})^2 + 1} < 1, \tag{4}$$

and may be further improved in most cases. The condition (4) is shown to be slightly stronger than the sufficient condition given in [2]. More importantly, however, the *method* in which it is derived allows us to generalize to many other situations:

- The condition (4) is easily improved for graphs with irregular geometry or potential strengths
- The method also provides a bound on the distance between any two BP fixed points.
- The same methodology may be applied to the case of quantized or otherwise approximated messages, yielding bounds on the ensuing error (our original motivation).
- By regarding message errors as a stochastic process and applying a few additional assumptions, a similar analysis obtains alternate, tighter estimates (though not necessarily bounds) of performance.

## 4  Message Approximations

In order to discuss the effects and propagation of errors introduced to the BP messages, we first require a measure of the difference between two messages. Although there are certainly other possibilities, it is very natural to consider the message deviations (which we denote $e_{ts}$) to be multiplicative, or additive in the log-domain, and examine a measure of the error's *dynamic range*:

$$\hat{m}_{ts}(x_s) = m_{ts}(x_s)e_{ts}(x_s) \qquad\qquad d(e_{ts}) = \max_{a,b} \sqrt{e_{ts}(a)/e_{ts}(b)} \tag{5}$$

Then, we have that $m_{ts}(x) = \hat{m}_{ts}(x)\forall x$ if and only if $\log d(e_{ts}) = 0$. This measure may also be related to more traditional error measures, including an absolute error on $\log m(x)$, a floating-point precision on $m(x)$, and the Kullback-Leibler divergence $D(m(x)\|\hat{m}(x))$; for details, see [14]. In this light our analysis of message approximation (Section 5.3) may be equivalently regarded as a statement about the required precision for an accurate implementation of loopy BP. Figure 2 shows an example message $m(x)$ and approximation $\hat{m}(x)$ along with their associated error $e(x)$.

To facilitate our analysis, we split the message update operation (2) into two parts. In the first, we focus on the message *products*

$$M_{ts}(x_t) \propto \psi_t(x_t) \prod_{u\in\Gamma_t\setminus s} m_{ut}(x_t) \qquad\qquad M_t(x_t) \propto \psi_t(x_t) \prod_{u\in\Gamma_t} m_{ut}(x_t) \tag{6}$$

where as usual, the proportionality constant is chosen to normalize $M$. We show the message error metric is (sub-)additive, i.e. that the errors in each incoming message (at most) add in their impact on $M$. The second operation is the message *convolution*

$$m_{ts}(x_s) \propto \int \psi_{ts}(x_t, x_s) M_{ts}(x_t) dx_t \tag{7}$$

where $M$ is a normalized message or product of messages. We demonstrate a level of *contraction*, that is, the approximation of $m_{ts}$ is measurably better than the approximation of $M_{ts}$ used to construct it.

We use the convention that lowercase quantities $(m_{ts}, e_{ts}, \ldots)$ refer to messages and message errors, while uppercase ones $(M_{ts}, E_{ts}, M_t, \ldots)$ refer to products of messages or errors—all incoming messages to node $t$ ($M_t$ and $E_t$), or all except the one from $s$ ($M_{ts}$ and $E_{ts}$). Due to space constraints, many omitted details and proofs can be found in [14].

## 4.1 Additivity and Error Contraction

The log of (5) is sub-additive, since for several incoming messages $\{\hat{m}_{ut}(x)\}$ we have

$$\log d(E_{ts}) = \log d\left(\hat{M}_{ts}/M_{ts}\right) = \log d\left(\prod e_{ut}\right) \leq \sum \log d(e_{ut}) \tag{8}$$

We may also derive a minimum rate of contraction on the errors. We consider the message from $t$ to $s$; since all quantities in this section relate to $m_{ts}$ and $M_{ts}$ we suppress the subscripts. The error measure $d(e)$ is given by

$$d(e)^2 = d(\hat{m}/m)^2 = \max_{a,b} \frac{\int \psi(x_t, a) M(x_t) E(x_t) dx_t}{\int \psi(x_t, a) M(x_t) dx_t} \cdot \frac{\int \psi(x_t, b) M(x_t) dx_t}{\int \psi(x_t, b) M(x_t) E(x_t) dx_t} \tag{9}$$

subject to certain constraints, such as positivity of the messages and potentials. Since

$$\forall f, g > 0, \qquad \int f(x)\, dx \,/\, \int g(x)\, dx \leq \max_x f(x)/g(x) \tag{10}$$

we can directly obtain the two bounds:

$$d(e)^2 \leq d(E)^2 \quad \text{and} \quad d(e)^2 \leq d(\psi)^4 \tag{11}$$

where we have extended the measure $d(\cdot)$ to functions of two variables (describing a minimum rate of *mixing* across the potential) by

$$d(\psi)^2 = \max_{a,b,c,d} \frac{\psi(a,b)}{\psi(c,d)}. \tag{12}$$

However, with some work one may show [14] the stronger measure of contraction,

$$d(e) \leq \frac{d(\psi)^2 d(E) + 1}{d(\psi)^2 + d(E)}. \tag{13}$$

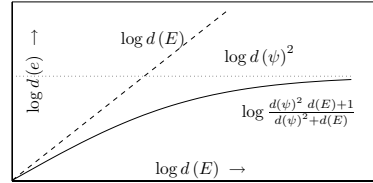

Figure 3: Bounds on the error output $d(e)$ as a function of the error in the product of incoming messages $d(E)$.

*Sketch of proof:* While the full proof is rather involved, we outline the procedure here. First, use (10) to show that the maximum of (9) given $d(\psi)$ is attained by potentials of the form $\psi(x, a) \propto 1 + K\chi_A(x)$ and $\psi(x, b) \propto 1 + K\chi_B(x)$, where $K = d(\psi)^2 - 1$ and $\chi_A$ and $\chi_B$ take on only values $\{0, 1\}$, along with a similar form for $E(x)$. Then define the variables $M_A = \int M(x)\zeta_A(x)$, $M_{AE} = \int M(x)\zeta_A(x)\zeta_E(x)$, etc., and optimize given the constraints $0 \leq M_A$, $M_B$, $M_E \leq 1$, $M_{AE} \leq \min[M_A, M_E]$, and $M_{BE} \geq \max[0, M_E - (1 - M_B)]$ (where the last constraint arises from the fact that $M_E + M_B - M_{BE} \leq 1$). Simplifying and taking the square root yields (13).

The bound (13) is shown in Figure 3; note that it improves both error bounds (11), shown as straight lines. In the next section, we use (8)-(13) to analyze the behavior of loopy BP.

## 5   Implications in Graphs with Cycles

We begin by examining loopy BP with exact message passing, using the previous results to derive a new sufficient condition for convergence to a unique fixed point. When this

condition is not satisfied, we instead obtain a bound on the relative distances between any two fixed points of the loopy BP equations. We then consider the effect of introducing additional errors into the messages passed at each iteration, showing sufficient conditions for this operation to converge, and a bound on the resulting error from exact loopy BP.

## 5.1 Convergence of Loopy BP & Fixed Point Distance

Tatikonda and Jordan [2] showed that the convergence and fixed points of loopy BP may be considered in terms of a Gibbs measure on the graph's computation tree, implying that loopy BP is guaranteed to converge if the graph satisfies Dobrushin's condition [15]. Dobrushin's condition is a global measure and difficult to verify; given in [2] is a sufficient condition (often called Simon's condition):

$$\max_t \sum_{u \in \Gamma_t} \log d\left(\psi_{ut}\right) < 1 \tag{14}$$

where $d\left(\psi\right)$ is defined as in (12). Using the previous section's analysis, we may argue something slightly stronger. Let us take the "true" messages $m_{ts}$ to be any fixed point of BP, and "approximate" them at each iteration by performing loopy BP from some arbitrary initial conditions. Now suppose that the largest message-product error $\log d\left(E_{ut}\right)$ in any node $u$ with parent $t$ at level $i$ of the computation tree (corresponding to iteration $n-i$ out of $n$ total iterations of loopy BP) is bounded above by some constant $\log \epsilon^i$. Note that this is trivially true (at any $i$) for the constant $\log \epsilon^i = \max_{(u,t)\in\mathcal{E}} |\Gamma_t| \log d\left(\psi_{ut}\right)^2$. Now, we may bound $d\left(E_{ts}\right)$ at any replicate of node $t$ with parent $s$ on level $i-1$ of the tree by

$$\log d\left(E_{ts}\right) \le g_{ts}(\log \epsilon^i) = \sum_{u \in \Gamma_t \setminus s} \log \frac{d\left(\psi_{ut}\right)^2 \epsilon^i + 1}{d\left(\psi_{ut}\right)^2 + \epsilon^i}. \tag{15}$$

and we may define $\log \epsilon^{i-1} = \max_{t,s} g_{ts}(\log \epsilon^i)$ to bound the error at level $i-1$. Loopy BP will converge if the sequence $\epsilon^i, \epsilon^{i-1}, \dots$ is strictly decreasing for all $\epsilon > 1$, i.e. $g_{ts}(z) < z$ for all $z > 0$. This is guaranteed by the conditions $g_{ts}(0) = 0$, $g'_{ts}(0) < 1$ and $g''_{ts}(z) < 0$. The first is easy to show, the third can be verified by algebra, and the condition $g'_{ts}(0) < 1$ can be rewritten to give the convergence criterion

$$\max_{(s,t)\in\mathcal{E}} \sum_{u \in \Gamma_t \setminus s} \frac{d\left(\psi_{ut}\right)^2 - 1}{d\left(\psi_{ut}\right)^2 + 1} < 1 \tag{16}$$

We may relate (16) to Simon's condition (14) by expanding the set $\Gamma_t \setminus s$ to the larger $\Gamma_t$ and noting that $\log x \ge \frac{x^2-1}{x^2+1}$ for all $x \ge 1$ with equality as $x \to 1$. Doing so, we see that Simon's condition is sufficient to guarantee (16), but that (16) may be true (implying convergence) when Simon's condition is not satisfied. The improvement over Simon's condition becomes negligible as connectivity increases (assuming the graph has approximately equal-strength potentials), but can be significant for low connectivity. For example, if the graph consists of a single loop then each node $t$ has at most two neighbors. In this case, the contraction (16) tells us that the outgoing message in either direction is *always* closer to the BP fixed point than the incoming message. Thus we obtain the result of [1], that (for finite-strength potentials) BP always converges to a unique fixed point on graphs containing a single loop. Simon's condition, on the other hand, is too loose to demonstrate this fact.

If the condition (16) is not satisfied, then the sequence $\{\epsilon^i\}$ is not always decreasing and there may be multiple fixed points. In this case, the sequence $\{\epsilon^i\}$ as defined will decrease until it reaches the largest value $\epsilon$ such that $\max_{ts} g_{ts}(\log \epsilon) = \log \epsilon$. Since the choice of initialization was arbitrary, we may opt to initialize to *any other* fixed point, and observe that the difference $E_t$ between these two fixed point beliefs is bounded by

$$\log d\left(E_t\right) \le \sum_{u \in \Gamma_t} \log \frac{d\left(\psi_{ut}\right)^2 \epsilon + 1}{d\left(\psi_{ut}\right)^2 + \epsilon} \tag{17}$$

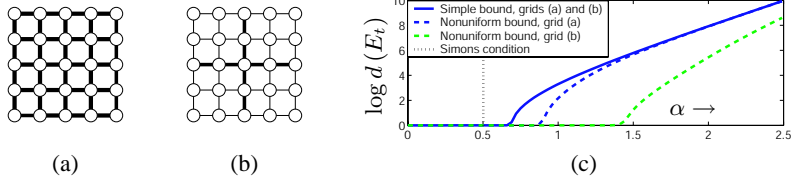

(a)            (b)                        (c)

Figure 4: Two small ($5 \times 5$) grids, with (a) all equal-strength potentials $\log d\left(\psi\right)^2 = \alpha$ and (b) several weaker ones ($\log d\left(\psi\right)^2 = .5\alpha$, thin lines). The methods described provide bounds (c) on the distance between any two fixed points as a function of potential strength $\alpha$, all of which improve on Simon's condition. See text for details.

Thus, the fixed points of BP lie in some potentially small set. If $\log \epsilon$ is small (the condition (16) is nearly satisfied) then although we cannot guarantee convergence to a unique fixed point, we can guarantee that every fixed point and our estimate are all mutually close (in a log-ratio sense).

## 5.2   Improving the Bounds by Path-counting

If we are willing to put a bit more effort into our bound-computation, we may be able to improve it.In particular the proofs of (16)-(17) assume that, as a message error propagates through the graph, repeated convolution with *only* the strongest set of potentials is possible. But often even if the worst potentials are quite strong, every cycle which contains them also contains several weaker potentials. Using an iterative algorithm much like BP itself, we may obtain a more globally aware estimate of error propagation.

Let us consider a message-passing procedure (potentially performed offline) in which node $t$ passes a (scalar) bound $\upsilon_{ts}^i$ on the message error $d\left(e_{ts}^i\right)$ at iteration $i$ to its neighbor $s$. The bound may be initialized to $\upsilon_{ts}^1 = d\left(\psi_{ts}\right)^2$, and the next iteration's (updated) outgoing bound is given by the pair of equations

$$\log \upsilon_{ts}^{i+1} = \log \frac{d\left(\psi_{ts}\right)^2 \epsilon_{ts}^i + 1}{d\left(\psi_{ts}\right)^2 + \epsilon_{ts}^i} \qquad\qquad \log \epsilon_{ts}^i = \sum_{u \in \Gamma_t \setminus s} \log \upsilon_{ut}^i \qquad (18)$$

Here, as in Section 5.1, $\epsilon_{ts}^i$ bounds the error $d\left(E_{ts}\right)$ in the product of incoming messages.

If (18) converges to $\log \upsilon_{ts}^i \to 0$ for all $t, s$ we may guarantee a unique fixed point for loopy BP; if not, we may compute $\log \epsilon_t^i = \sum_{\Gamma_t} \log \upsilon_{ut}^i$ to obtain a bound on the belief error at each node $t$. If every node is identical (same number of neighbors, same potential strengths) this yields the same bound as (17); however, if the graph or potential strengths are inhomogeneous it provides a strictly stronger bound on loopy BP convergence and errors.

This situation is illustrated in Figure 4—we specify two $5 \times 5$ grids in terms of their potential strengths and compute bounds on the log-range of their fixed point beliefs. (While potential strength does not completely specify the graphical model, it is sufficient for all the bounds considered here.) One grid (a) has equal-strength potentials $\log d\left(\psi\right)^2 = \alpha$, while the other has many weaker potentials ($\alpha/2$). The worst-case bounds are the same (since both have a node with four strong neighbors), shown as the solid curve in (c). However, the dashed curves show the estimate of (18), which improves only slightly for the strongly coupled graph (a) but considerably for the weaker graph (b). All three bounds give considerably more information than Simon's condition (dotted vertical line).

## 5.3   Introducing additional errors

As discussed in the introduction, we may wish to introduce or allow *additional* errors in our messages at each stage, in order to improve the computational or communication efficiency of the algorithm. This may be the result of an actual distortion imposed on the message

(perhaps to decrease its complexity, for example quantization), from censoring the message update (reusing the message from the previous iteration) when the two are sufficiently similar, or from approximating or quantizing the model parameters (potential functions). Any of these additional errors can be easily incorporated into our framework.

If at each iteration, we introduce an additional (perhaps stochastic) error to each message which has a dynamic range bounded by some constant $\delta$, the relationships of (18) become

$$\log v_{ts}^{i+1} = \log \frac{d\left(\psi_{ts}\right)^2 \epsilon_{ts}^i + 1}{d\left(\psi_{ts}\right)^2 + \epsilon_{ts}^i} + \log \delta \qquad \log \epsilon_{ts}^i = \sum_{u \in \Gamma_t \backslash s} \log v_{ut}^i \qquad (19)$$

and gives a bound on the steady-state error (distance from a fixed point) in the system.

### 5.4 Stochastic Analysis

Unfortunately, the above bounds are often pessimistic compared to actual performance. By treating the perturbations as stochastic we may obtain a more realistic estimate (though no longer a strict bound) on the resulting error. Specifically, let us describe the error functions $\log e_{ts}(x_s)$ for each $x_s$ as a random variable with mean zero and variance $\sigma_{ts}^2$. By assuming that the errors in each incoming message are uncorrelated, we obtain additivity of their variances: $\Sigma_{ts}^2 = \sum_{u \in \Gamma_t \backslash s} \sigma_{ut}^2$. The assumption of uncorrelated errors is clearly questionable since propagation around loops may couple the incoming message errors, but is common in quantization analysis, and we shall see that it appears reasonable in practice.

We would also like to estimate the contraction of variance incurred in the convolution step. We may do so by applying a simple sigma-point quadrature ("unscented") approximation [16], in which the standard deviation of the convolved function $m_{ts}(x_s)$ is estimated by applying the same nonlinearity (13) to the standard deviation of the error on the incoming product $M_{ts}$. Thus, similarly to (18) and (19), we have

$$\sigma_{ts}^2 = \left( \log \frac{d\left(\psi_{ts}\right)^2 \lambda_{ts} + 1}{\lambda_{ts} + d\left(\psi_{ts}\right)^2} \right)^2 + (\log \delta)^2 \qquad (\log \lambda_{ts})^2 = \sum_{u \in \Gamma_t \backslash s} \sigma_{ut}^2 \qquad (20)$$

The steady-state solution of (20) yields an estimate of the variances of the log-belief $\log p_t$ by $\sigma_t^2 = \sum_{u \in \Gamma_t} \sigma_{ut}^2$; this estimate is typically much smaller than the bound (18) due to the strict sub-additive relationship between the standard deviations. Although it is *not* a bound, using a Chebyshev-like argument we may conclude that, for example, the $2\sigma_t$ distance will be greater than the typical errors observed in practice.

## 6 Experiments

We demonstrate the error bounds for perturbed messages with a set of Monte Carlo trials. In particular, for each trial we construct a binary-valued $5 \times 5$ grid with uniform potential strengths, which are either (1) all positively correlated, or (2) randomly chosen to be positively or negatively correlated (equally likely); we also assign random single-node potentials to each $x_s$. We then run a quantized version of BP, rounding each log-message to discrete values separated by $2 \log \delta$ (ensuring that the newly introduced error satisifies $d\left(e\right) \leq \delta$). Figure 5 shows the maximum belief error in each of 100 trials of this procedure for various values of $\delta$.

Also shown are the bound on belief error developed in Section 5.3 and the $2\sigma$ estimate computed assuming uncorrelated message errors. As can be seen, the stochastic estimate is often a much tighter, more accurate assessment of error, but it does not possess the same strong theoretical guarantees. Since, as observed in analysis of quantization and stability in digital filtering [17], the errors introduced by quantization are typically close to independent, the assumptions of the stochastic estimate are reasonable and empirically we observe that the estimate and actual errors behave similarly.

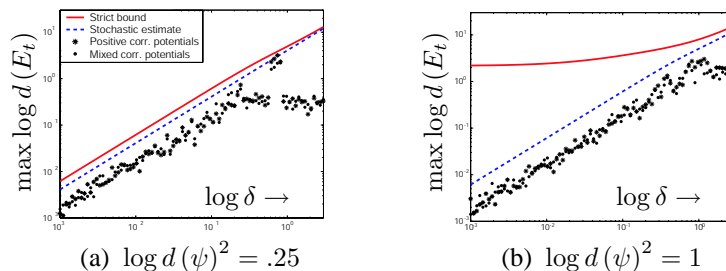

$$\text{(a)} \quad \log d\left(\psi\right)^2 = .25 \qquad\qquad \text{(b)} \quad \log d\left(\psi\right)^2 = 1$$

Figure 5: Maximum belief errors incurred as a function of the quantization error. The scatterplot indicates the maximum error measured in the graph for each of 200 Monte Carlo runs; this is strictly bounded above by the solution of (18), solid, and bounded with high probability (assuming uncorrelated errors) by (20), dashed.

## 7   Conclusions

We have described a particular measure of distortion on BP messages and shown that it is sub-additive and measurably contractive, leading to sufficient conditions for loopy BP to converge to a unique fixed point. Furthermore, this enables analysis of quantized, stochastic, or other approximate forms of BP, yielding sufficient conditions for convergence and bounds on the deviation from exact message passing. Assuming the perturbations are uncorrelated can often give tighter estimates of the resulting error. For additional details as well as some further consequences and extensions, see [14].

*The authors would like to thank Erik Sudderth, Martin Wainwright, Tom Heskes, and Lei Chen for many helpful discussions. This research was supported in part by MIT Lincoln Laboratory under Lincoln Program 2209-3023 and by ODDR&E MURI through ARO grant DAAD19-00-0466.*

## References

[1]  Y. Weiss. Correctness of local probability propagation in graphical models with loops. *Neural Computation*, 12(1), 2000.

[2]  S. Tatikonda and M. Jordan. Loopy belief propagation and gibbs measures. In *UAI*, 2002.

[3]  T. Heskes. On the uniqueness of loopy belief propagation fixed points. To appear in *Neural Computation*, 2004.

[4]  D. Koller, U. Lerner, and D. Angelov. A general algorithm for approximate inference and its application to hybrid Bayes nets. In *UAI 15*, pages 324–333, 1999.

[5]  J. M. Coughlan and S. J. Ferreira. Finding deformable shapes using loopy belief propagation. In *ECCV 7*, May 2002.

[6]  E. B. Sudderth, A. T. Ihler, W. T. Freeman, and A. S. Willsky. Nonparametric belief propagation. In *CVPR*, 2003.

[7]  M. Isard. PAMPAS: Real–valued graphical models for computer vision. In *CVPR*, 2003.

[8]  T. Minka. Expecatation propagation for approximate bayesian inference. In *UAI*, 2001.

[9]  A. T. Ihler, J. W. Fisher III, and A. S. Willsky. Communication-constrained inference. Technical Report TR-2601, Laboratory for Information and Decision Systems, 2004.

[10]  L. Chen, M. Wainwright, M. Cetin, and A. Willsky. Data association based on optimization in graphical models with application to sensor networks. Submitted to *Mathematical and Computer Modeling*, 2004.

[11]  P. Clifford. Markov random fields in statistics. In G. R. Grimmett and D. J. A. Welsh, editors, *Disorder in Physical Systems*, pages 19–32. Oxford University Press, Oxford, 1990.

[12]  J. Pearl. *Probabilistic Reasoning in Intelligent Systems*. Morgan Kaufman, San Mateo, 1988.

[13]  J. S. Yedidia, W. T. Freeman, and Y. Weiss. Constructing free energy approximations and generalized belief propagation algorithms. Technical Report 2004-040, MERL, May 2004.

[14]  A. T. Ihler, J. W. Fisher III, and A. S. Willsky. Message errors in belief propagation. Technical Report TR-2602, Laboratory for Information and Decision Systems, 2004.

[15]  Hans-Otto Georgii. *Gibbs measures and phase transitions*. Studies in Mathematics. de Gruyter, Berlin / New York, 1988.

[16]  S. Julier and J. Uhlmann. A general method for approximating nonlinear transformations of probability distributions. Technical report, RRG, Dept. of Eng. Science, Univ. of Oxford, 1996.

[17]  A. Willsky. Relationships between digital signal processing and control and estimation theory. *Proc. IEEE*, 66(9):996–1017, September 1978.